# A recipe for optimizing a time-histogram

**Hideaki Shimazaki**
Department of Physics, Graduate School of Science
Kyoto University
Kyoto 606-8502, Japan
shimazaki@ton.scphys.kyoto-u.ac.jp

**Shigeru Shinomoto**
Department of Physics, Graduate School of Science
Kyoto University
Kyoto 606-8502, Japan
shinomoto@scphys.kyoto-u.ac.jp

## Abstract

The time-histogram method is a handy tool for capturing the instantaneous rate of spike occurrence. In most of the neurophysiological literature, the bin size that critically determines the goodness of the fit of the time-histogram to the underlying rate has been selected by individual researchers in an unsystematic manner. We propose an objective method for selecting the bin size of a time-histogram from the spike data, so that the time-histogram best approximates the unknown underlying rate. The resolution of the histogram increases, or the optimal bin size decreases, with the number of spike sequences sampled. It is notable that the optimal bin size diverges if only a small number of experimental trials are available from a moderately fluctuating rate process. In this case, any attempt to characterize the underlying spike rate will lead to spurious results. Given a paucity of data, our method can also suggest how many more trials are needed until the set of data can be analyzed with the required resolution.

## 1 Introduction

The rate of spike occurrence, or the firing rate, of a neuron can be captured by the (peri-stimulus) time-histogram (PSTH) [1, 2], which is constructed easily as follows: Align spike sequences to the onset of stimuli, divide time into discrete bins, count the number of spikes that enter each bin, and divide the counts by the bin size and the number of sequences. The shape of a PSTH depends on the choice of the bin size. With too large a bin size, one cannot represent the detailed time-dependent rate, while with too small a bin size, the time-histogram fluctuates greatly and one cannot discern the underlying spike rate. There exists an ideal bin size for estimating the spike rate for each set of experimental data. This important parameter has mostly been selected subjectively by individual researchers.

We devised a method of selecting the bin size objectively so that a PSTH best approximates the underlying rate, which is unknown. In the course of our study, we found an interesting paper that proposed an empirical method of choosing the histogram bin size for a probability density function (Rudemo M, (1982) *Scandinavian Journal of Statistics* **9**: 65-78 [3]). Although applicable to a Poisson point process, this theory appears to have rarely been applied to PSTHs. It would be preferable to have a theory in accordance with the procedures of neurophysiological experiments in which a stimulus is repeated to extract a signal from a neuron. Given a set of experimental data, we wish to

not only determine the optimal bin size, but also estimate how many more experimental trials should be performed in order to obtain a resolution we deem sufficient.

It was revealed by a theoretical analysis that the optimal bin size may diverge for a small number of spike sequences derived from a moderately fluctuating rate [4]. This implies that any attempt to characterize the underlying rate will lead to spurious results. The present method can indicate the divergence of the optimal bin size only from the spike data. Even under such a condition, the present method nevertheless provides an inference on the number of trails that need to be performed in order to obtain a meaningful estimated rate.

## 2  Methods

We consider sequences of spikes repeatedly recorded from identical experimental trials. A recent analysis revealed that *in vivo* spike trains are not simply random, but possess inter-spike-interval distributions intrinsic and specific to individual neurons [5, 6]. However, spikes accumulated from a large number of spike trains recorded from a single neuron are, in the majority, mutually independent. Being free from the intrinsic inter-spike-interval distributions of individual spike trains, the accumulated spikes can be regarded as being derived repeatedly from Poisson processes of an identical time-dependent rate [7, 8].

It would be natural to assess the goodness of the fit of the estimator $\hat{\lambda}_t$ to the underlying spike rate $\lambda_t$ over the total observation period $T$ by the mean integrated squared error (MISE),

$$\text{MISE} \equiv \frac{1}{T} \int_0^T E\left(\hat{\lambda}_t - \lambda_t\right)^2 dt, \tag{1}$$

where $E$ refers to the expectation over different realization of point events, given $\lambda_t$. We suggest a method for minimizing the MISE with respect to the bin size $\Delta$. The difficulty of the present problem comes from the fact that the underlying spike rate $\lambda_t$ is not known.

### 2.1  Selection of the bin size

We choose the (bar-graph) PSTH as a way to estimate the rate $\hat{\lambda}_t$, and explore a method to select the bin size of a PSTH that minimizes MISE in Eq.(1). A PSTH is constructed simply by counting the number of spikes that belong to each bin. For an observation period $T$, we obtain $N = \lfloor T/\Delta \rfloor$ intervals. The number of spikes accumulated from all $n$ sequences in the $i$th interval is counted as $k_i$. The bar height at the $i$th bin is given by $k_i/n\Delta$.

Given a bin of width $\Delta$, the expected height of a bar graph for $t \in [0, \Delta]$ is the time-averaged rate,

$$\theta = \frac{1}{\Delta} \int_0^\Delta \lambda_t \, dt. \tag{2}$$

The total number of spikes $k$ from $n$ spike sequences that enter a bin of width $\Delta$ obeys a Poisson distribution with the expected number $n\Delta\theta$,

$$p(k \,|\, n\Delta\theta) = \frac{(n\Delta\theta)^k}{k!} e^{-n\Delta\theta}. \tag{3}$$

The unbiased estimator for $\theta$ is given as $\hat{\theta} = k/(n\Delta)$, which is the empirical height of the bar graph for $t \in [0, \Delta]$.

By segmenting the total observation period $T$ into $N$ intervals of size $\Delta$, the MISE defined in Eq.(1) can be rewritten as

$$\text{MISE} = \frac{1}{\Delta} \int_0^\Delta \frac{1}{N} \sum_{i=1}^N \left\{ E\left(\hat{\theta}_i - \lambda_{t+(i-1)\Delta}\right)^2 \right\} dt, \tag{4}$$

where $\hat{\theta}_i \equiv k_i/(n\Delta)$. Hereafter we denote the average over those segmented rate $\lambda_{t+(i-1)\Delta}$ as an average over an ensemble of (segmented) rate functions $\{\lambda_t\}$ defined in an interval of $t \in [0, \Delta]$:

$$\text{MISE} = \frac{1}{\Delta} \int_0^\Delta \left\langle E\left(\hat{\theta} - \lambda_t\right)^2 \right\rangle dt. \tag{5}$$

| (i) | Divide the observation period $T$ into $N$ bins of width $\Delta$, and count the number of spikes $k_i$ from all $n$ sequences that enter the $i$th bin. |
|---|---|
| (ii) | Construct the mean and variance of the number of spikes $\{k_i\}$ as, $$\bar{k} \equiv \frac{1}{N}\sum_{i=1}^{N} k_i, \text{ and } v \equiv \frac{1}{N}\sum_{i=1}^{N}(k_i - \bar{k})^2.$$ |
| (iii) | Compute the cost function, $$C_n(\Delta) = \frac{2\bar{k} - v}{(n\Delta)^2}.$$ |
| (iv) | Repeat i through iii while changing the bin size $\Delta$ to search for $\Delta^*$ that minimizes $C_n(\Delta)$. |

The expectation $E$ now refers to the average over the spike count, or $\hat{\theta} = k/(n\Delta)$, given a rate function $\lambda_t$, or its mean value, $\theta$. The MISE can be decomposed into two parts,

$$\text{MISE} = \frac{1}{\Delta}\int_0^\Delta \left\langle E\left(\hat{\theta} - \theta + \theta - \lambda_t\right)^2 \right\rangle dt = \left\langle E(\hat{\theta} - \theta)^2 \right\rangle + \frac{1}{\Delta}\int_0^\Delta \left\langle (\lambda_t - \theta)^2 \right\rangle dt. \qquad (6)$$

The first and second terms are respectively the stochastic fluctuation of the estimator $\hat{\theta}$ around the expected mean rate $\theta$, and the temporal fluctuation of $\lambda_t$ around its mean $\theta$ over an interval of length $\Delta$, averaged over the segments.

The second term of Eq.(6) can further be decomposed into two parts,

$$\frac{1}{\Delta}\int_0^\Delta \left\langle (\lambda_t - \langle\theta\rangle + \langle\theta\rangle - \theta)^2 \right\rangle dt = \frac{1}{\Delta}\int_0^\Delta \left\langle (\lambda_t - \langle\theta\rangle)^2 \right\rangle dt - \left\langle (\theta - \langle\theta\rangle)^2 \right\rangle. \qquad (7)$$

The first term in the rhs of Eq.(7) represents a mean squared fluctuation of the underlying rate $\lambda_t$ from the mean rate $\langle\theta\rangle$, and is independent of the bin size $\Delta$, because

$$\frac{1}{\Delta}\int_0^\Delta \left\langle (\lambda_t - \langle\theta\rangle)^2 \right\rangle dt = \frac{1}{T}\int_0^T (\lambda_t - \langle\theta\rangle)^2 \; dt. \qquad (8)$$

We define a cost function by subtracting this term from the original MISE,

$$\begin{aligned} C_n(\Delta) &\equiv \text{MISE} - \frac{1}{\Delta}\int_0^\Delta \left\langle (\lambda_t - \langle\theta\rangle)^2 \right\rangle dt \\ &= \left\langle E(\hat{\theta} - \theta)^2 \right\rangle - \left\langle (\theta - \langle\theta\rangle)^2 \right\rangle. \end{aligned} \qquad (9)$$

This cost function corresponds to the "risk function" in the report by Rudemo, (Eq. 2.3), obtained by direct decomposition of the MISE [3]. The second term in Eq.(9) represents the temporal fluctuation of the expected mean rate $\theta$ for individual intervals of period $\Delta$. As the expected mean rate is not an observable quantity, we must replace the fluctuation of the expected mean rate with that of the observable estimator $\hat{\theta}$. Using the decomposition rule for an unbiased estimator ($E\hat{\theta} = \theta$),

$$\left\langle E(\hat{\theta} - \langle E\hat{\theta}\rangle)^2 \right\rangle = \left\langle E(\hat{\theta} - \theta + \theta - \langle\theta\rangle)^2 \right\rangle = \left\langle E(\hat{\theta} - \theta)^2 \right\rangle + \left\langle (\theta - \langle\theta\rangle)^2 \right\rangle, \qquad (10)$$

the cost function is transformed into

$$C_n(\Delta) = 2\left\langle E(\hat{\theta} - \theta)^2 \right\rangle - \left\langle E(\hat{\theta} - \langle E\hat{\theta}\rangle)^2 \right\rangle. \qquad (11)$$

Due to the assumed Poisson nature of the point process, the number of spikes $k$ counted in each bin obeys a Poisson distribution: the variance of $k$ is equal to the mean. For the estimated rate defined as $\hat{\theta} = k/(n\Delta)$, this variance-mean relation corresponds to

$$E(\hat{\theta} - \theta)^2 = \frac{1}{n\Delta} E\hat{\theta}. \tag{12}$$

By incorporating Eq.(12) into Eq.(11), the cost function is given as a function of the estimator $\hat{\theta}$,

$$C_n (\Delta) = \frac{2}{n\Delta} \left\langle E\hat{\theta} \right\rangle - \left\langle E(\hat{\theta} - \langle E\hat{\theta} \rangle)^2 \right\rangle. \tag{13}$$

The optimal bin size is obtained by minimizing the cost function $C_n(\Delta)$:

$$\Delta^* \equiv \arg\min_{\Delta} C_n(\Delta). \tag{14}$$

By replacing the expectation of $\hat{\theta}$ in Eq.(13) with the sample spike counts, the method is converted into a user-friendly recipe summarized in Table 1.

## 2.2 Extrapolation of the cost function

With the method developed in the preceding subsection, we can determine the optimal bin size for a given set of experimental data. In this section, we develop a method to estimate how the optimal bin size decreases when more experimental trials are added to the data set.

Assume that we are in possession of $n$ spike sequences. The fluctuation of the expected mean rate $\left\langle (\theta - \langle\theta\rangle)^2 \right\rangle$ in Eq.(10) is replaced with the empirical fluctuation of the time-histogram $\hat{\theta}_n$ using the decomposition rule for the unbiased estimator $\hat{\theta}_n$ satisfying $E\hat{\theta}_n = \theta$,

$$\left\langle E(\hat{\theta}_n - \langle E\hat{\theta}_n \rangle)^2 \right\rangle = \left\langle E(\hat{\theta}_n - \theta + \theta - \langle\theta\rangle)^2 \right\rangle = \left\langle E(\hat{\theta}_n - \theta)^2 \right\rangle + \left\langle (\theta - \langle\theta\rangle)^2 \right\rangle. \tag{15}$$

The expected cost function for $m$ sequences can be obtained by substituting the above equation into Eq.(9), yielding

$$C_m (\Delta|n) = \left\langle E(\hat{\theta}_m - \theta)^2 \right\rangle + \left\langle E(\hat{\theta}_n - \theta)^2 \right\rangle - \left\langle E(\hat{\theta}_n - \langle E\hat{\theta}_n \rangle)^2 \right\rangle. \tag{16}$$

Using the variance-mean relation for the Poisson distribution, Eq.(12), and

$$E(\hat{\theta}_m - \theta)^2 = \frac{1}{m\Delta} E\hat{\theta}_m = \frac{1}{m\Delta} E\hat{\theta}_n, \tag{17}$$

we obtain

$$C_m (\Delta|n) = \left( \frac{1}{m} - \frac{1}{n} \right) \frac{1}{\Delta} \left\langle E\hat{\theta}_n \right\rangle + C_n (\Delta), \tag{18}$$

where $C_n (\Delta)$ is the original cost function, Eq.(13), computed using the estimators $\hat{\theta}_n$. By replacing the expectation with sample spike count averages, the cost function for $m$ sequences can be extrapolated as $C_m (\Delta|n)$ with this formula, using the sample mean $\bar{k}$ and variance $v$ of the numbers of spikes, given $n$ sequences and the bin size $\Delta$. The extrapolation method is summarized in Table 2.

It may come to pass that the original cost function $C_n(\Delta)$ computed for $n$ spike sequences does not have a minimum, or have a minimum at a bin size comparable to the observation period $T$. In such a case, with the method summarized in Table 2, one may estimate the critical number of sequences $n_c$ above which the cost function has a finite bin size $\Delta^*$, and consider carrying out more experiments to obtain a reasonable rate estimation. In the case that the optimal bin size exhibits continuous divergence, the cost function can be expanded as

$$C_n(\Delta) \sim \mu \left( \frac{1}{n} - \frac{1}{n_c} \right) \frac{1}{\Delta} + u\frac{1}{\Delta^2}, \tag{19}$$

where we have introduced $n_c$ and $u$, which are independent of $n$. The optimal bin size undergoes a phase transition from the vanishing $1/\Delta^*$ for $n < n_c$ to a finite $1/\Delta^*$ for $n > n_c$. In this case, the inverse optimal bin size is expanded in the vicinity of $n_c$ as $1/\Delta^* \propto (1/n - 1/n_c)$. We can

Table 2: A method for extrapolating the cost function for a PSTH

(A)   Construct the extrapolated cost function,
$$C_m\left(\Delta|n\right) = \left(\frac{1}{m} - \frac{1}{n}\right)\frac{\bar{k}}{n\Delta^2} + C_n(\Delta),$$
using the sample mean $\bar{k}$ and variance $v$ of the number of spikes
obtained from $n$ sequences of spikes.

(B)   Search for $\Delta_m^*$ that minimizes $C_m\left(\Delta|n\right)$.

(C)   Repeat A and B while changing $m$, and plot $1/\Delta_m^*$ vs $1/m$ to search for
the critical value $1/m = 1/\hat{n}_c$ above which $1/\Delta_m^*$ practically vanishes.

estimate the critical value $\hat{n}_c$ by applying this asymptotic relation to the set of $\hat{\Delta}_m^*$ estimated from $C_m(\Delta|n)$ for various values of $m$:

$$\frac{1}{\Delta_m^*} \propto \left(\frac{1}{m} - \frac{1}{\hat{n}_c}\right). \tag{20}$$

It should be noted that there are cases that the optimal bin size exhibits discontinuous divergence from a finite value. Even in such cases, the plot of $\{1/m, 1/\Delta^*\}$ could be useful in exploring a discontinuous transition from nonvanishing values of $1/\Delta^*$ to practically vanishing values.

## 2.3   Theoretical cost function

In this section, we obtain a "theoretical" cost function directly from a process with a known underlying rate, $\lambda_t$, and compare it with the "empirical" cost function which can be evaluated without knowing the rate process. Note that this theoretical cost function is not available in real experimental conditions in which the underlying rate is not known.

The present estimator $\hat{\theta} \equiv k/(n\Delta)$ is a uniformly minimum variance unbiased estimator (UMVUE) of $\theta$, which achieves the lower bound of the Cramér-Rao inequality [9, 10],

$$E(\hat{\theta} - \theta)^2 = \left[-\sum_{k=0}^{\infty} p\left(k|\theta\right)\frac{\partial^2 \log p\left(k|\theta\right)}{\partial \theta^2}\right]^{-1} = \frac{\theta}{n\Delta}. \tag{21}$$

Inserting this into Eq.(9), the cost function is represented as

$$\begin{aligned}
C_n\left(\Delta\right) &= \frac{\langle\theta\rangle}{n\Delta} - \left\langle\left(\theta - \langle\theta\rangle\right)^2\right\rangle \\
&= \frac{\mu}{n\Delta} - \frac{1}{\Delta^2}\int_0^\Delta\int_0^\Delta \phi\left(t_1 - t_2\right)dt_1 dt_2,
\end{aligned} \tag{22}$$

where $\mu$ is the mean rate, and $\phi(t)$ is the autocorrelation function of the rate fluctuation, $\lambda_t - \mu$. Based on the symmetry $\phi(t) = \phi(-t)$, the cost function can be rewritten as

$$\begin{aligned}
C_n\left(\Delta\right) &= \frac{\mu}{n\Delta} - \frac{1}{\Delta^2}\int_{-\Delta}^\Delta (\Delta - |t|)\phi(t)\,dt \\
&\approx \frac{\mu}{n\Delta} - \frac{1}{\Delta}\int_{-\infty}^\infty \phi(t)\,dt + \frac{1}{\Delta^2}\int_{-\infty}^\infty |t|\phi(t)\,dt,
\end{aligned} \tag{23}$$

which can be identified with Eq.(19) with parameters given by

$$n_c = \mu\left/\int_{-\infty}^\infty \phi(t)\,dt\right. , \tag{24}$$

$$u = \int_{-\infty}^\infty |t|\phi(t)\,dt. \tag{25}$$

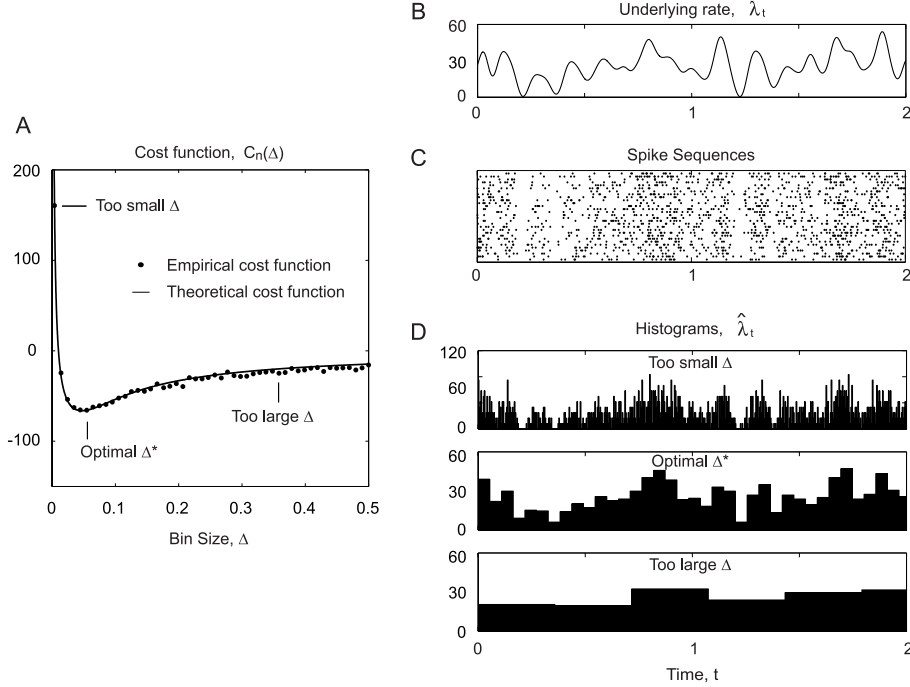

Figure 1: A: (Dots): The empirical cost function, $C_n(\Delta)$, computed from spike data according to the method in Table 1. (Solid line): The "theoretical" cost function computed directly from the underlying fluctuating rate, with Eq.(22). B: (Above): The underlying fluctuating rate $\lambda_t$. (Middle): Spike sequences derived from the rate. (Below): Time-histograms made using three types of bin sizes: too small, optimal, and too large. Model parameters: the number of sequences $n = 30$; total observation period $T = 30$ [sec]; the mean rate $\mu = 30$ [1/s]; the amplitude of rate fluctuation $\sigma = 10$ [1/s]; time scale of rate fluctuation $\tau = 0.05$ [s].

## 3  Results

Our first objective was to develop a method for selecting the ideal bin size using spike sequences derived repeatedly from Poisson processes, all with a given identical rate $\lambda_t$. The MISE of the PSTH from the underlying rate is minimized by minimizing the cost function $C_n(\Delta)$. Figure 1A displays the cost function computed with the method summarized in Table 1. This "empirical" cost function is compared with the "theoretical" cost function Eq.(22) that is computed directly from the underlying rate $\lambda_t$. The figure exhibits that the "empirical" cost function is consistent with the "theoretical" cost function. The time-histogram constructed using the optimal bin size is compared with those constructed using non-optimal bin sizes in Figs. 1B, demonstrating the effectiveness of the present method of bin size selection.

We also tested a method for extrapolating the cost function. Figures 2A and B demonstrate the extrapolated cost functions for several sequences with differing values of $m$ and the plot of $\{1/m, 1/\Delta^*\}$ for estimating the critical value $1/m = 1/\hat{n}_c$, above which $1/\Delta^*$ practically vanishes. Figure 2C depicts the critical number $\hat{n}_c$ estimated from the smaller or larger numbers of spike sequences $n$. The empirically estimated critical number $\hat{n}_c$ approximates the theoretically predicted critical number $n_c$ computed using Eq.(24). Note that the critical number is correctly estimated from the small number of sequences, with which the optimal bin size practically diverges ($n < n_c$).

## 4  Summary

We have developed a method for optimizing the bin size, so that the PSTH best represents the (unknown) underlying spike rate. For a small number of spike sequences derived from a modestly

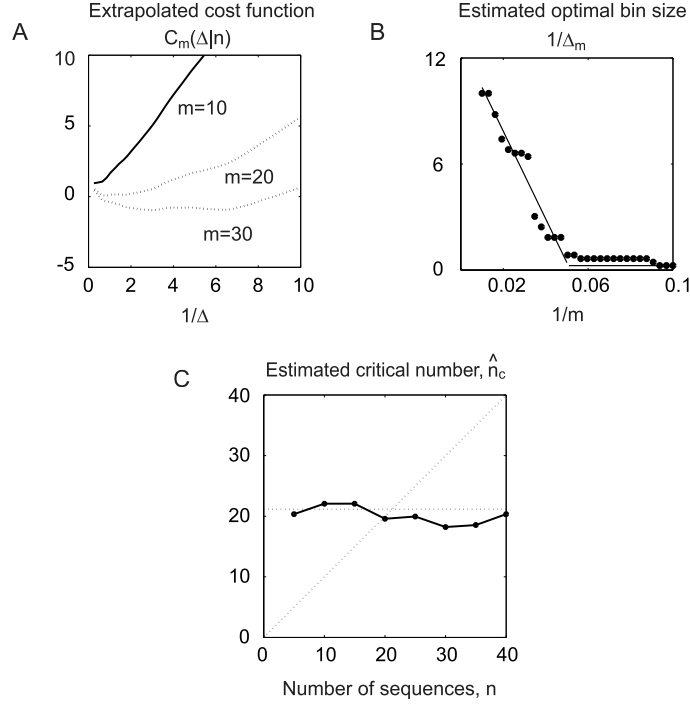

Figure 2: A: Extrapolated cost functions $C_m\left(\Delta|n\right)$ plotted against $1/\Delta$ for several numbers of sequences $m = 10, 20$ and $30$ computed from $n = 10$ sample sequences. B: The plot of $\{1/m, 1/\Delta^*\}$ used for estimating the critical value $1/m = 1/\hat{n}_c$, above which $1/\Delta^*$ practically vanishes. C: The number of spike sequences $n$ used to obtain the extrapolated cost function $C_m\left(\Delta|n\right)$ and an estimated critical number $\hat{n}_c$. Model parameters: the number of sequences $n = 10$; total observation period $T = 30$ [sec]; the mean rate $\mu = 30$ [1/s]; the amplitude of rate fluctuation $\sigma = 4$ [1/s]; time scale of rate fluctuation $\tau = 0.05$ [s]. The theoretical critical number is computed with Eq.(24), giving $n_c = 21.1$ for the present underlying fluctuating rate. This theoretical $n_c$ is depicted as the horizontal dashed line.

fluctuating rate, the cost function does not have a minimum, implying the uselessness of the rate estimation. Our method can nevertheless extrapolate the cost function for any number of spike sequences, and suggest how many trials are needed in order to obtain a meaningful time-histogram with the required accuracy. The suitability of the present method was demonstrated by application to spike sequences generated by time-dependent Poisson processes.

**Acknowledgements**

This study is supported in part by Grants-in-Aid for Scientific Research to SS from the Ministry of Education, Culture, Sports, Science and Technology of Japan (16300068, 18020015) and the 21st Century COE "Center for Diversity and Universality in Physics". HS is supported by the Research Fellowship of the Japan Society for the Promotion of Science for Young Scientists.

# References

[1] E. D. Adrian. *The Basis of Sensation: The Action of the Sense Organs*. W.W. Norton, New York, 1928.

[2] G. L. Gerstein and N. Y. S. Kiang. An approach to the quantitative analysis of electrophysiological data from single neurons. *Biophysical Journal*, 1(1):15–28, 1960.

[3] M. Rudemo. Empirical choice of histograms and kernel density estimators. *Scandinavian Journal of Statistics*, 9(2):65–78, 1982.

[4] S. Koyama and S. Shinomoto. Histogram bin width selection for time-dependent poisson processes. *Journal of Physics A-Mathematical and General*, 37(29):7255–7265, 2004.

[5] S. Shinomoto, K. Shima, and J. Tanji. Differences in spiking patterns among cortical neurons. *Neural Computation*, 15(12):2823–2842, 2003.

[6] S. Shinomoto, Y. Miyazaki, H. Tamura, and I. Fujita. Regional and laminar differences in in vivo firing patterns of primate cortical neurons. *Journal of Neurophysiology*, 94(1):567–575, 2005.

[7] D. L. Snyder. *Random Point Processes*. John Wiley & Sons, Inc., New York, 1975.

[8] D. J. Daley and D. Vere-Jones. *An Introduction to the Theory of Point Processes*. Springer-Verlag, New York, USA, 1988.

[9] R. E. Blahut. *Principles and practice of information theory*. Addison-Wesley, Reading, Mass, 1987.

[10] T. M. Cover and J. A. Thomas. *Elements of Information Theory*. John Wiley & Sons, Inc., New York, 1991.
